# Learning Patient-Specific Cancer Survival Distributions as a Sequence of Dependent Regressors

**Chun-Nam Yu, Russell Greiner, Hsiu-Chin Lin**
Department of Computing Science
University of Alberta
Edmonton, AB T6G 2E8
{chunnam,rgreiner,hsiuchin}@ualberta.ca

**Vickie Baracos**
Department of Oncology
University of Alberta
Edmonton, AB T6G 1Z2
vickie.baracos@ualberta.ca

## Abstract

An accurate model of patient survival time can help in the treatment and care of cancer patients. The common practice of providing survival time estimates based only on population averages for the site and stage of cancer ignores many important individual differences among patients. In this paper, we propose a local regression method for learning patient-specific survival time distribution based on patient attributes such as blood tests and clinical assessments. When tested on a cohort of more than 2000 cancer patients, our method gives survival time predictions that are much more accurate than popular survival analysis models such as the Cox and Aalen regression models. Our results also show that using patient-specific attributes can reduce the prediction error on survival time by as much as 20% when compared to using cancer site and stage only.

## 1 Introduction

When diagnosed with cancer, most patients ask about their prognosis: "how long will I live", and "what is the success rate of each treatment option". Many doctors provide patients with statistics on cancer survival based only on the site and stage of the tumor. Commonly used statistics include the 5-year survival rate and median survival time, e.g., a doctor can tell a specific patient with early stage lung cancer that s/he has a 50% 5-year survival rate.

In general, today's cancer survival rates and median survival times are estimated from a large group of cancer patients; while these estimates do apply to the population in general, they are not particularly accurate for individual patients, as they do not include patient-specific information such as age and general health conditions. While doctors can make adjustments to their survival time predictions based on these individual differences, it is better to directly incorporate these important factors explicitly in the prognostic models – e.g. by incorporating the clinical information, such as blood tests and performance status assessments [1] that doctors collect during the diagnosis and treatment of cancer. These data reveal important information about the state of the immune system and organ functioning of the patient, and therefore are very useful for predicting how well a patient will respond to treatments and how long s/he will survive. In this work, we develop machine learning techniques to incorporate this wealth of healthcare information to learn a more accurate prognostic model that uses patient-specific attributes. With improved prognostic models, cancer patients and their families can make more informed decisions on treatments, lifestyle changes, and sometimes end-of-life care.

In survival analysis [2], the Cox proportional hazards model [3] and other parametric survival distributions have long been used to fit the survival time of a population. Researchers and clinicians usually apply these models to compare the survival time of two populations or to test for significant risk factors affecting survival; n.b., these models are not designed for the task of predicting survival

time for individual patients. Also, as these models work with the hazard function instead of the survival function (see Section 2), they might not give good calibrated predictions on survival rates for individuals. In this work we propose a new method, *multi-task logistic regression* (MTLR), to learn patient-specific survival distributions. MTLR directly models the survival function by combining multiple local logistic regression models in a dependent manner. This allows it to handle censored observations and the time-varying effects of features naturally. Compared to survival regression methods such as the Cox and Aalen regression models, MTLR gives significantly more accurate predictions on survival rates over several datasets, including a large cohort of more than 2000 cancer patients. MTLR also reduces the prediction error on survival time by 20% when compared to the common practice of using the median survival time based on cancer site and stage.

Section 2 surveys basic survival analysis and related works. Section 3 introduces our method for learning patient-specific survival distributions. Section 4 evaluates our learned models on a large cohort of cancer patients, and also provides additional experiments on two other datasets.

## 2 Survival Time Prediction for Cancer Patients

In most regression problems, we know both the covariates and "outcome" values for all individuals. By contrast, it is typical to *not* know many of the outcome values in survival data. In many medical studies, the event of interest for many individuals (death, disease recurrence) might not have occurred within the fixed period of study. In addition, other subjects could move out of town or decide to drop out any time. Here we know only the date of the final visit, which provides a lower bound on the survival time. We refer to the time recorded as the "event time", whether it is the true survival time, or just the time of the last visit (censoring time). Such datasets are considered *censored*.

Survival analysis provides many tools for modeling the survival time $T$ of a population, such as a group of stage-3 lung cancer patients. A basic quantity of interest is the survival function $S(t) = P(T \geq t)$, which is the probability that an individual within the population will survive longer than time $t$. Given the survival times of a set of individuals, we can plot the proportion of surviving individuals against time, as a way to visualize $S(t)$. The plot of this empirical survival distribution is called the Kaplan-Meier curve [4] (Figure 1(left)).

This is closely related to the hazard function $\lambda(t)$, which describes the instantaneous rate of failure at time $t$

$$\lambda(t) = \lim_{\Delta t \to 0} P(t \leq T < t + \Delta t \mid T \geq t)/\Delta t, \text{ and } S(t) = \exp\left(-\int_0^t \lambda(u)du\right).$$

### 2.1 Regression Models in Survival Analysis

One of the most well-known regression model in survival analysis is Cox's proportional hazards model [3]. It assumes the hazard function $\lambda(t)$ depends multiplicatively on a set of features $\vec{x}$:

$$\lambda(t \mid \vec{x}) = \lambda_0(t) \exp(\vec{\theta} \cdot \vec{x}).$$

It is called the *proportional hazards model* because the hazard rates of two individuals with features $\vec{x}_1$ and $\vec{x}_2$ differ by a ratio $\exp(\vec{\theta} \cdot (\vec{x}_1 - \vec{x}_2))$. The function $\lambda_0(t)$, called the baseline hazard, is usually left unspecified in Cox regression. The regression coefficients $\vec{\theta}$ are estimated by maximizing a partial likelihood objective, which depends only on the relative ordering of survival time of individuals but not on their actual values. Cox regression is mostly used for identifying important risk factors associated with survival in clinical studies. It is typically not used to predict survival time since the hazard function is incomplete without the baseline hazard $\lambda_0$. Although we can fit a non-parametric survival function for $\lambda_0(t)$ after the coefficients of Cox regression are determined [2], this requires a cumbersome 2-step procedure. Another weakness of the Cox model is its proportional hazards assumption, which restricts the effect of each feature on survival to be constant over time.

There are alternatives to the Cox model that avoids the proportional hazards restriction, including the Aalen additive hazards model [5] and other time-varying extensions to the Cox model [6]. The Aalen linear hazard model assumes the hazard function has the form

$$\lambda(t \mid \vec{x}) = \vec{\theta}(t) \cdot \vec{x}. \tag{1}$$

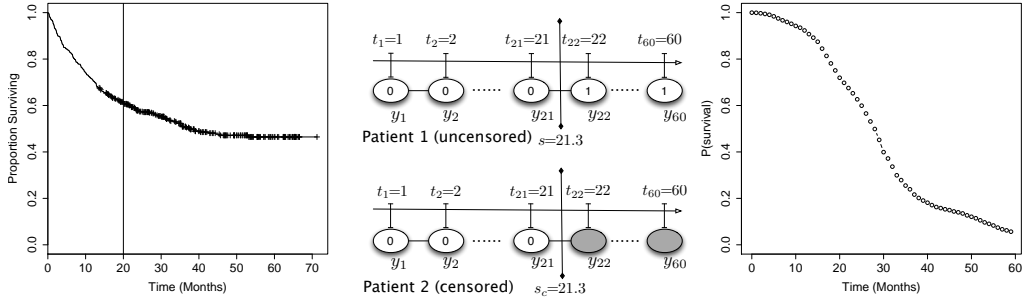

Figure 1: (Left) Kaplan-Meier curve: each point $(x, y)$ means proportion $y$ of the patients are alive at time $x$. Vertical line separates those who have died versus those who survive at $t = 20$ months. (Middle) Example binary encoding for patient 1 (uncensored) with survival time 21.3 months and for patient 2 (censored), with last visit time at 21.3 months. (Right) Example discrete survival function for a single patient predicted by MTLR.

While there are now many estimation techniques, goodness-of-fit tests, hypothesis tests for these survival regression models, they are rarely evaluated on the task of predicting survival time of individual patients. Moreover, it is not easy to choose between the various assumptions imposed by these models, such as whether the hazard rate should be a multiplicative or additive function of the features. In this paper we will test our MTLR method, which directly models the survival function, against Cox regression and Aalen regression as representatives of these survival analysis models.

In machine learning, there are a few recently proposed regression technqiues for survival prediction [7, 8, 9, 10]. These methods attempt to optimize specific loss function or performance measures, which usually involve modifying the common regression loss functions to handle censored data. For example, Shivaswamy et al. [7] modified the *support vector regression* (SVR) loss function from

$$\max\left\{|y - \vec{\theta} \cdot \vec{x}| - \epsilon, 0\right\} \quad \text{to} \quad \max\left\{(y - \vec{\theta} \cdot \vec{x}) - \epsilon, 0\right\},$$

where $y$ is the time of censoring and $\epsilon$ is a tolerance parameter. In this way any prediction $\vec{\theta} \cdot \vec{x}$ above the censoring time $y$ is deemed consistent with observation and is not penalized. This class of direct regression methods usually give very good results on the particular loss functions they optimize over, but could fail if the loss function is non-convex or difficult to optimize. Moreover, these methods only predict a single survival time value (a real number) without an associated confidence on prediction, which is a serious drawback in clinical applications.

Our MTLR model below is closely related to local regression models [11] and varying coefficient models [12] in statistics. Hastie and Tibshirani [12] described a very general class of regression models that allow the coefficients to change with another set of variables called "effect modifiers"; they also discussed an application of their model to overcome the proportional hazards assumption in Cox models. While we focus on predicting survival time, they instead focused on evaluating the time-varying effect of prognostic factors and worked with the rank-based partial likelihood objective.

## 3   Survival Distribution Modeling via a Sequence of Dependent Regressors

Consider a simpler classification task of predicting whether an individual will survive for more than $t$ months. A common approach for this classification task is the logistic regression model [13], where we model the probability of surviving more than $t$ months as:

$$P_{\vec{\theta}}(T \geq t \mid \vec{x}) = \left(1 + \exp(\vec{\theta} \cdot \vec{x} + b)\right)^{-1}.$$

The parameter vector $\vec{\theta}$ describes the effect of how the features $\vec{x}$ affect the chance of survival, with the threshold $b$. This task corresponds to a specific time point on the Kaplan-Meier curve, which attempts to discriminate those who survive against those who have died, based on the features $\vec{x}$ (Figure 1(left)). Equivalently, the logistic regression model can be seen as modeling the individual survival probabilities of cancer patients at the time snapshot $t$.

Taking this idea one step further, consider modeling the probability of survival of patients at each of a vector of time points $\tau = (t_1, t_2, \ldots, t_m)$ – e.g., $\tau$ could be the 60 monthly intervals from 1 month

up to 60 months. We can set up a series of logistic regression models for each of these:

$$P_{\vec{\theta}_i}(T \geq t_i \mid \vec{x}) = \left(1 + \exp(\vec{\theta}_i \cdot \vec{x} + b_i)\right)^{-1}, \qquad 1 \leq i \leq m, \qquad (2)$$

where $\vec{\theta}_i$ and $b_i$ are time-specific parameter vector and thresholds. The input features $\vec{x}$ stay the same for all these classification tasks, but the binary labels $y_i = [T \geq t_i]$ can change depending on the threshold $t_i$. This particular setup allows us to answer queries about the survival probability of individual patients at each of the time snapshots $\{t_i\}$, getting close to our goal of modeling a personal survival time distribution for individual patients. The use of time-specific parameter vector naturally allows us to capture the effect of time-varying covariates, similar to many dynamic regression models [14, 12].

However the outputs of these logistic regression models are not independent, as a death event at or before time $t_i$ implies death at all subsequent time points $t_j$ for all $j > i$. MTLR enforces the dependency of the outputs by predicting the survival status of a patient at each of the time snapshots $t_i$ jointly instead of independently. We encode the survival time $s$ of a patient as a binary sequence $y = (y_1, y_2, \ldots, y_m)$, where $y_i \in \{0, 1\}$ denotes the survival status of the patient at time $t_i$, so that $y_i = 0$ (no death event yet) for all $i$ with $t_i < s$, and $y_i = 1$ (death) for all $i$ with $t_i \geq s$ (see Figure 1(middle)). We denote such an encoding of the survival time $s$ as $y(s)$, and let $y_i(s)$ be the value at its $i$th position. Here there are $m + 1$ possible legal sequences of the form $(0, 0, \ldots, 1, 1, \ldots, 1)$, including the sequence of all '0's and the sequence of all '1's. The probability of observing the survival status sequence $y = (y_1, y_2, \ldots, y_m)$ can be represented by the following generalization of the logistic regression model:

$$P_{\Theta}(Y = (y_1, y_2, \ldots, y_m) \mid \vec{x}) = \frac{\exp(\sum_{i=1}^{m} y_i(\vec{\theta}_i \cdot \vec{x} + b_i))}{\sum_{k=0}^{m} \exp(f_{\Theta}(\vec{x}, k))},$$

where $\Theta = (\vec{\theta}_1, \ldots, \vec{\theta}_m)$, and $f_{\Theta}(\vec{x}, k) = \sum_{i=k+1}^{m}(\vec{\theta}_i \cdot \vec{x} + b_i)$ for $0 \leq k \leq m$ is the score of the sequence with the event occuring in the interval $[t_k, t_{k+1})$ before taking the logistic transform, with the boundary case $f_{\Theta}(\vec{x}, m) = 0$ being the score for the sequence of all '0's. This is similar to the objective of conditional random fields [15] for sequence labeling, where the labels at each node are scored and predicted jointly.

Therefore the log likelihood of a set of uncensored patients with survival time $s_1, s_2, \ldots, s_n$ and feature vectors $\vec{x}_1, \vec{x}_2, \ldots, \vec{x}_n$ is

$$\sum_{i=1}^{n} \left[ \sum_{j=1}^{m} y_j(s_i)(\vec{\theta}_j \cdot \vec{x}_i + b_j) - \log \sum_{k=0}^{m} \exp f_{\Theta}(\vec{x}_i, k) \right].$$

Instead of directly maximizing this log likelihood, we solve the following optimization problem:

$$\min_{\Theta} \frac{C_1}{2} \sum_{j=1}^{m} \|\vec{\theta}_j\|^2 + \frac{C_2}{2} \sum_{j=1}^{m-1} \|\vec{\theta}_{j+1} - \vec{\theta}_j\|^2 - \sum_{i=1}^{n} \left[ \sum_{j=1}^{m} y_j(s_i)(\vec{\theta}_j \cdot \vec{x}_i + b_j) - \log \sum_{k=0}^{m} \exp f_{\Theta}(\vec{x}_i, k) \right] \quad (3)$$

The first regularizer over $\|\vec{\theta}_j\|^2$ ensures the norm of the parameter vector is bounded to prevent overfitting. The second regularizer $\|\vec{\theta}_{j+1} - \vec{\theta}_j\|^2$ ensures the parameters vary smoothly across consecutive time points, and is especially important for controlling the capacity of the model when the time points become dense. The regularization constants $C_1$ and $C_2$, which control the amount of smoothing for the model, can be estimated via cross-validation. As the above optimization problem is convex and differentiable, optimization algorithms such as Newton's method or quasi-Newton methods can be applied to solve it efficiently. Since we model the survival distribution as a series of dependent prediction tasks, we call this model *multi-task logistic regression* (MTLR). Figure 1(right) shows an example survival distribution predicted by MTLR for a test patient.

### 3.1  Handling Censored Data

Our multi-task logistic regression model can handle censoring naturally by marginalizing over the unobserved variables in a survival status sequence $(y_1, y_2, \ldots, y_m)$. For example, suppose a patient with features $\vec{x}$ is censored at time $s_c$, and $t_j$ is the closest time point after $s_c$. Then all the sequences

Table 1: Left: number of cancer patients for each site and stage in the cancer registry dataset. Right: features used in learning survival distributions

| site\stage | 1 | 2 | 3 | 4 | basic | age, sex, weight gain/loss, BMI, cancer site, cancer stage |
|---|---|---|---|---|---|---|
| Bronchus & Lung | 61 | 44 | 186 | 390 | | |
| Colorectal | 15 | 157 | 233 | 545 | general wellbeing | no appetite, nausea, sore mouth, |
| Head and Neck | 6 | 8 | 14 | 206 | | taste funny, constipation, pain, |
| Esophagus | 0 | 1 | 1 | 63 | | dental problem, dry mouth, vomit, |
| Pancreas | 1 | 3 | 0 | 134 | | diarrhea, performance status |
| Stomach | 0 | 0 | 1 | 128 | blood test | granulocytes, LDH-serum, HGB, |
| Other Digestive | 0 | 1 | 0 | 77 | | lyphocytes platelet, WBC count, |
| Misc | 1 | 0 | 3 | 123 | | calcium-serum, creatinine, albumin |

$y = (y_1, y_2, \ldots, y_m)$ with $y_i = 0$ for $i < j$ are consistent with this censored observation (see Figure 1(middle)). The likelihood of this censored patient is

$$P_\Theta(T \geq t_j \mid \vec{x}) = \sum\nolimits_{k=j}^{m} \exp(f_\Theta(\vec{x}, k)) / \sum\nolimits_{k=0}^{m} \exp(f_\Theta(\vec{x}, k)), \quad (4)$$

where the numerator is the sum over all consistent sequences. While the sum in the numerator makes the log-likelihood non-concave, we can still learn the parameters effectively using EM or gradient descent with suitable initialization.

In summary, the proposed MTLR model holds several advantages over classical regression models in survival analysis for survival time prediction. First, it directly models the more intuitive survival function rather than the hazard function (conditional rate of failure/death), avoiding the difficulties of choosing between different forms of hazards. Second, by modeling the survival distribution as the joint output of a sequence of dependent local regressors, we can capture the time-varying effects of features and handle censored data easily and naturally. Third, we will see that our model can give more accurate predictions on survival and better calibrated probabilities (see Section 4), which are important in clinical applications.

Our goal here is not to replace these tried-and-tested models in survival analysis, which are very effective for hypothesis testing and prognostic factor discovery. Instead, we want a tool that can accurately and effectively predict an individual's survival time.

### 3.2   Relations to Other Machine Learning Models

The objective of our MTLR model is of the same form as a general CRF [15], but there are several important differences from typical applications of CRFs for sequence labeling. First MTLR has no transition features (edge potentials) (Eq (3)); instead the dependencies between labels in the sequence are enforced implicitly by only allowing a linear number ($m+1$) of legal labelings. Second, in most sequence labeling applications of CRFs, the weights for the node potentials are shared across nodes to share statistic strengths and improve generalization. Instead, MTLR uses a different weight vector $\vec{\theta}_i$ at each node to capture the time-varying effects of input features. Unlike typical sequence labeling problems, the sequence construction of our model might be better viewed as a device to obtain a flexible discrete approximation of the survival distribution of individual patients.

Our approach can also be seen as an instance of multi-task learning [16], where the prediction of individual survival status at each time snapshot $t_j$ can be regarded as a separate task. The smoothing penalty $\|\vec{\theta}_j - \vec{\theta}_{j+1}\|^2$ is used by many multi-task regularizers to encourage weight sharing between related tasks. However, unlike typical multi-task learning problems, in our model the outputs of different tasks are dependent to satisfy the monotone condition of a survival function.

## 4   Experiments

Our main dataset comes from the Alberta Cancer Registry obtained through the Cross Cancer Institute at the University of Alberta, which included 2402 cancer patients with tumors at different sites. About one third of the patients have censored survival times. Table 1 shows the groupings of cancer patients in the dataset and the patient-specific attributes for learning survival distributions. All these measurements are taken before the first chemotherapy.

In all experiments we report five-fold cross validation (5CV) results, where MTLR's regularization parameters $C_1$ and $C_2$ are selected by another 5CV within the training fold, based on log likelihood. We pick the set of time points $\tau$ in these experiments to be the 100 points from the 1st percentile up to the 100th percentile of the event time (true survival time or censoring time) over all patients. Since all the datasets contain censored data, we first train an MTLR model using the event time (survival/censoring) as regression targets (no hidden variables). Then the trained model is used as the initial weights in the EM procedure in Eq (4) to train the final model.

The Cox proportional hazards model is trained using the `survival` package in R, followed by the fitting of the baseline hazard $\lambda_0(t)$ using the Kalbfleisch-Prentice estimator [2]. The Aalen linear hazards model is trained using the `timereg` package. Both the Cox and the Aalen models are trained using the same set of 25 features. As a baseline for this cancer registry dataset, we also provide a prediction based on the median survival time and survival probabilities of the subgroup of patients with cancer at a specific site and at a specific stage, estimated from the training fold.

## 4.1 Survival Rate Prediction

Our first evaluation focuses on the classification accuracy and calibration of predicted survival probabilities at different time thresholds. In addition to giving a binary prediction on whether a patient would survive beyond a certain time period, say 2 years, it is very useful to give an associated confidence of the prediction in terms of probabilities (survival rate). We use mean square error (MSE), also called the Brier score in this setting [17], to measure the quality of probability predictions. Previous work [18] showed that MSE can be decomposed into two components, one measuring calibration and one measuring discriminative power (i.e., classification accuracy) of the probability predictions.

Table 2 shows the classification accuracy and MSE on the predicted probabilities of different models at 5, 12, and 22 months, which correspond to the 25% lower quantile, median, and 75% upper quantile of the survival time of all the cancer patients in the dataset. Our MTLR models produce predictions on survival status and survival probability that are much more accurate than the Cox and Aalen regression models. This shows the advantage of directly modeling the survival function instead of going through the hazard function when predicting survival probabilites. The Cox model and the Aalen model have classification accuracies and MSE that are similar to one another on this dataset. All regression models (MTLR, Cox, Aalen) beat the baseline prediction using median survival time based on cancer stage and site only, indicating that there is substantial advantage of employing extra clinical information to improve survival time predictions given to cancer patients.

## 4.2 Visualization

Figure 2 visualizes the MTLR, Cox and Aalen regression models for two patients on a test fold. Patient 1 is a short survivor who lives for only 3 months from diagnosis, while patient 2 is a long survivor whose survival time is censored at 46 months. All three regression models (correctly) give poor prognosis for patient 1 and good prognosis for patient 2, but there are a few interesting differences when we examine the plots. The MTLR model is able to produce smooth survival curves of different shapes for the two patients (one convex with the other one slightly concave), while the Cox model always predict survival curves of similar shapes because of the proportional hazards assumption. Indeed it is well known that the survival curves of two individuals never crosses for a Cox model. For the Aalen model, we observe that the survival function is not (locally) monotonically decreasing. This is a consequence of the linear hazards assumption (Eq (1)), which allows the hazard to become negative and therefore the survival function to increase. This problem is less common when predicting survival curves at population level, but could be more frequent for individual survival distribution predictions.

## 4.3 Survival Time Predictions Optimizing Different Loss Functions

Our third evaluation on the predicted survival distributions involves applying them to make predictions that minimize different clinically-relevant loss functions. For example, if the patient is interested in knowing whether s/he has weeks, months, or years to live, then measuring errors in terms of the *logarithm* of the survival time can be appropriate. In this case we can measure the loss

Table 2: Classification accuracy and MSE of survival probability predictions on cancer registry dataset (standard error of 5CV shown in brackets). **Bold** numbers indicate significance with a paired t-test at $p = 0.05$ level (this applies to all subsequent tables).

| Accuracy | 5 month | 12 month | 22 month | MSE | 5 month | 12 month | 22 month |
|----------|---------|----------|----------|-----|---------|----------|----------|
| MTLR | **86.5** (0.7) | **76.1** (0.9) | **74.5** (1.3) | MTLR | **0.101** (0.005) | **0.158** (0.004) | **0.170** (0.007) |
| Cox | 74.5 (0.9) | 59.3 (1.1) | 62.8 (3.5) | Cox | 0.196 (0.009) | 0.270 (0.008) | 0.232 (0.016) |
| Aalen | 73.3 (1.2) | 61.0 (1.7) | 59.6 (3.6) | Aalen | 0.198 (0.004) | 0.278 (0.008) | 0.288 (0.020) |
| Baseline | 69.2 (0.3) | 56.2 (2.0) | 57.0 (1.4) | Baseline | 0.227 (0.012) | 0.299 (0.011) | 0.243 (0.012) |

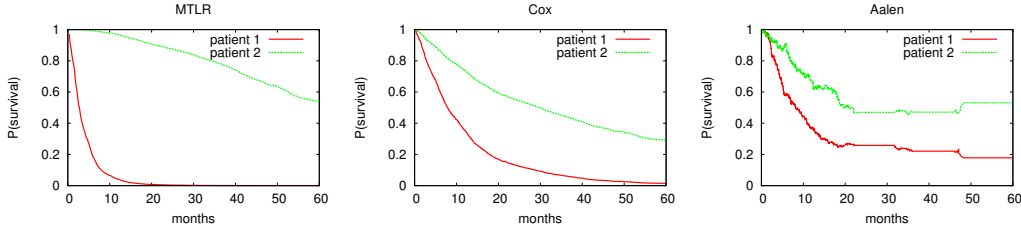

Figure 2: Predicted survival function for two patients in test set: MTLR (left), Cox (center), Aalen (right). Patient 1 lives for 3 months while patient 2 has survival time censored at 46 months.

using the absolute error (AE) over log survival time

$$l_{\text{AE}-\log}(p, t) = |\log p - \log t|, \tag{5}$$

where $p$ and $t$ are the predicted and true survival time respectively.

In other scenarios, we might be more concerned about the difference of the predicted and true survival time. For example, as the cost of hospital stays and medication scales linearly with the survival time, the AE loss on the survival time could be appropriate, i.e,

$$l_{\text{AE}}(p, t) = |p - t|. \tag{6}$$

We also consider an error measure called the *relative absolute error* (RAE):

$$l_{\text{RAE}}(p, t) = \min \left\{ |(p - t)/p|, 1 \right\}, \tag{7}$$

which is essentially AE scaled by the predicted survival time $p$, since $p$ is known at prediction time in clinical applications. The loss is truncated at 1 to prevent large penalizations for small predicted survival time. Knowing that the average RAE of a predictor is 0.3 means we can expect the true survival time to be within 30% of the predicted time.

Given any of these loss models $l$ above, we can make a point prediction $h_l(\vec{x})$ of the survival time for a patient with features $\vec{x}$ using the survival distribution $P_\Theta$ estimated by our MTLR model:

$$h_l(\vec{x}) = \operatorname*{argmin}_{p \in \{t_1, \ldots, t_m\}} \sum_{k=0}^{m} l(p, t_k) P_\Theta(Y = y(t_k) \mid \vec{x}), \tag{8}$$

where $y(t_k)$ is the survival time encoding defined in Section 3.

Table 3 shows the results on optimizing the three proposed loss functions using the individual survival distribution learned with MTLR against other methods. For this particular evaluation, we also implemented the censored support vector regression (CSVR) proposed in [7, 8]. We train two CSVR models, one using the survival time and the other using logarithm of the survival time as regression targets, which correspond to minimizing the AE and AE-log loss functions. For RAE we report the best result from linear and log-scale CSVR in the table, since this non-convex loss is not minimized by either of them. As we do not know the true survival time for censored patients, we adopt the approach of not penalizing a prediction $p$ for a patient with censoring time $t$ if $p > t$, i.e., $l(p, t) = 0$ for the loss functions defined in Eqs (5) to (7) above. This is exactly the same censored training loss used in CSVR. Note that it is undesirable to test on uncensored patients only, as the survival time distributions are very different for censored and uncensored patients. For Cox and Aalen models we report results using predictions based on the median, as optimizing for different loss functions using Eq (8) with the distributions predicted by Cox and Aalen models give inferior results.

The results in Table 3 show that, although CSVR has the advantage of optimizing the loss function directly during training, our MTLR model is still able to make predictions that improve on CSVR,

Table 3: Results on Optimizing Different Loss Functions on the Cancer Registry Dataset

|        | MTLR | Cox | Aalen | CSVR | Baseline |
|--------|------|-----|-------|------|----------|
| AE     | **9.58** (0.11) | 10.76 (0.12) | 19.06 (2.04) | **9.96** (0.32) | 11.73 (0.62) |
| AE-log | **0.56** (0.02) | 0.61 (0.02) | 0.76 (0.06) | **0.56** (0.02) | 0.70 (0.05) |
| RAE    | **0.40** (0.01) | 0.44 (0.02) | 0.44 (0.02) | 0.44 (0.03) | 0.53 (0.02) |

Table 4: (Top) MSE of Survival Probability Predictions on SUPPORT2 (left) and RHC (right). (Bottom) Results on Optimizing Different Loss Functions: SUPPORT2 (left), RHC (right)

| Support2 | 14 day | 58 day | 252 day | RHC | 8 day | 27 day | 163 day |
|----------|--------|--------|---------|-----|-------|--------|---------|
| MTLR  | **0.102**(0.002) | **0.162**(0.002) | 0.189(0.004) | MTLR  | **0.121**(0.002) | **0.175**(0.005) | **0.201**(0.004) |
| Cox   | 0.152(0.003) | 0.213(0.004) | 0.199(0.006) | Cox   | 0.180(0.005) | 0.239(0.004) | 0.223(0.004) |
| Aalen | 0.141(0.003) | 0.195(0.004) | 0.195(0.008) | Aalen | 0.176(0.004) | 0.229(0.006) | 0.221(0.006) |

| Support2 | AE | AE-log | RAE | RHC | AE | AE-log | RAE |
|----------|----|--------|-----|-----|----|--------|-----|
| MTLR  | **11.74** (0.35) | **1.19** (0.03) | **0.53** (0.01) | MTLR  | **2.90** (0.09) | 1.07 (0.02) | **0.49** (0.01) |
| Cox   | 14.08 (0.49) | 1.35 (0.03) | 0.71 (0.01) | Cox   | 3.08 (0.09) | 1.10 (0.02) | 0.53 (0.01) |
| Aalen | 14.61 (0.66) | 1.28 (0.04) | 0.65 (0.01) | Aalen | 3.55 (0.85) | 1.10 (0.06) | 0.54 (0.01) |
| CSVR  | **11.62** (0.15) | **1.18** (0.02) | 0.65 (0.01) | CSVR  | **2.96** (0.07) | 1.09 (0.02) | 0.58 (0.01) |

sometimes significantly. Moreover MTLR is able to make survival time prediction with improved RAE, which is difficult for CSVR to optimize directly. MTLR also beats the Cox and Aalen models on all three loss functions. When compared to the baseline of predicting the median survival time by cancer site and stage, MTLR is able to employ extra clinical features to reduce the absolute error on survival time from 11.73 months to 9.58 months, and the error ratio between true and predicted survival time from being off by $\exp(0.70) \approx 2.01$ times to $\exp(0.56) \approx 1.75$ times. Both error measures are reduced by about 20%.

## 4.4 Evaluation on Other Datasets

As additional evaluations, we also tested our model on the SUPPORT2 and RHC datasets (available at `http://biostat.mc.vanderbilt.edu/wiki/Main/DataSets`), which record the survival time for patients hospitalized with severe illnesses. SUPPORT2 contains over 9000 patients (32% censored) while RHC contains over 5000 patients (35% censored).

Table 4 (top) shows the MSE on survival probability prediction over the SUPPORT2 dataset and RHC dataset (we omit classification accuracy due to lack of space). The thresholds are again chosen at 25% lower quantile, median, and 75% upper quantile of the population survival time. The MTLR model, again, produces significantly more accurate probabilty predictions when compared against the Cox and Aalen regression models. Table 4 (bottom) shows the results on optimizing different loss functions for SUPPORT2 and RHC. The results are consistent with the cancer registry dataset, with MTLR beating Cox and Aalen regressions while tying with CSVR on AE and AE-log.

## 5 Conclusions

We plan to extend our model to an online system that can update survival predictions with new measurements. Our current data come from measurements taken when cancers are first diagnosed; it would be useful to be able to update survival predictions for patients incrementally, based on new blood tests or physician's assessments.

We have presented a new method for learning patient-specific survival distributions. Experiments on a large cohort of cancer patients show that our model gives much more accurate predictions of survival rates when compared to the Cox or Aalen survival regression models. Our results demonstrate that incorporating patient-specific features can significantly improve the accuracy of survival prediction over just using cancer site and stage, with prediction errors reduced by as much as 20%.

**Acknowledgments**

This work is supported by Alberta Innovates Centre for Machine Learning (AICML) and NSERC. We would also like to thank the Alberta Cancer Registry for the datasets used in this study.

# References

[1] M.M. Oken, R.H. Creech, D.C. Tormey, J. Horton, T.E. Davis, E.T. McFadden, and P.P. Carbone. Toxicity and response criteria of the eastern cooperative oncology group. *American Journal of Clinical Oncology*, 5(6):649, 1982.

[2] J.D. Kalbfleisch and R.L. Prentice. *The statistical analysis of failure time data*. Wiley New York:, 1980.

[3] D.R. Cox. Regression models and life-tables. *Journal of the Royal Statistical Society. Series B (Methodological)*, 34(2):187–220, 1972.

[4] E.L. Kaplan and P. Meier. Nonparametric estimation from incomplete observations. *Journal of the American Statistical Association*, 53(282):457–481, 1958.

[5] O.O. Aalen. A linear regression model for the analysis of life times. *Statistics in Medicine*, 8(8):907–925, 1989.

[6] T. Martinussen and T.H. Scheike. *Dynamic regression models for survival data*. Springer Verlag, 2006.

[7] P.K. Shivaswamy, W. Chu, and M. Jansche. A support vector approach to censored targets. In *ICDM 2007*, pages 655–660. IEEE, 2008.

[8] A. Khosla, Y. Cao, C.C.Y. Lin, H.K. Chiu, J. Hu, and H. Lee. An integrated machine learning approach to stroke prediction. In *KDD*, pages 183–192. ACM, 2010.

[9] V. Raykar, H. Steck, B. Krishnapuram, C. Dehing-Oberije, and P. Lambin. On ranking in survival analysis: Bounds on the concordance index. *NIPS*, 20, 2007.

[10] G.C. Cawley, N.L.C. Talbot, G.J. Janacek, and M.W. Peck. Sparse bayesian kernel survival analysis for modeling the growth domain of microbial pathogens. *IEEE Transactions on Neural Networks*, 17(2):471–481, 2006.

[11] W.S. Cleveland and S.J. Devlin. Locally weighted regression: an approach to regression analysis by local fitting. *Journal of the American Statistical Association*, 83(403):596–610, 1988.

[12] T. Hastie and R. Tibshirani. Varying-coefficient models. *Journal of the Royal Statistical Society. Series B (Methodological)*, 55(4):757–796, 1993.

[13] B. Efron. Logistic regression, survival analysis, and the Kaplan-Meier Curve. *Journal of the American Statistical Association*, 83(402):414–425, 1988.

[14] D. Gamerman. Dynamic Bayesian models for survival data. *Applied Statistics*, 40(1):63–79, 1991.

[15] J. Lafferty, A. McCallum, and F. Pereira. Conditional random fields: Probabilistic models for segmenting and labeling sequence data. In *ICML*, pages 282–289, 2001.

[16] R. Caruana. Multitask learning. *Machine Learning*, 28(1):41–75, 1997.

[17] G.W. Brier. Verification of forecasts expressed in terms of probability. *Monthly weather review*, 78(1):1–3, 1950.

[18] M.H. DeGroot and S.E. Fienberg. The comparison and evaluation of forecasters. *Journal of the Royal Statistical Society. Series D (The Statistician)*, 32(1):12–22, 1983.

